# Imitation Learning by Coaching

**He He    Hal Daumé III**
Department of Computer Science
University of Maryland
College Park, MD 20740
{hhe,hal}@cs.umd.edu

**Jason Eisner**
Department of Computer Science
Johns Hopkins University
Baltimore, MD 21218
jason@cs.jhu.edu

## Abstract

Imitation Learning has been shown to be successful in solving many challenging real-world problems. Some recent approaches give strong performance guarantees by training the policy iteratively. However, it is important to note that these guarantees depend on how well the policy we found can imitate the oracle on the training data. When there is a substantial difference between the oracle's ability and the learner's policy space, we may fail to find a policy that has low error on the training set. In such cases, we propose to use a *coach* that demonstrates easy-to-learn actions for the learner and gradually approaches the oracle. By a reduction of learning by demonstration to online learning, we prove that coaching can yield a lower regret bound than using the oracle. We apply our algorithm to cost-sensitive dynamic feature selection, a hard decision problem that considers a user-specified accuracy-cost trade-off. Experimental results on UCI datasets show that our method outperforms state-of-the-art imitation learning methods in dynamic feature selection and two static feature selection methods.

## 1   Introduction

Imitation learning has been successfully applied to a variety of applications [1, 2]. The standard approach is to use supervised learning algorithms and minimize a *surrogate loss* with respect to an oracle. However, this method ignores the difference between distributions of states induced by executing the oracle's policy and the learner's, thus has a quadratic loss in the task horizon $T$. A recent approach called Dataset Aggregation [3] (DAgger) yields a loss linear in $T$ by iteratively training the policy in states induced by all previously learned policies. Its theoretical guarantees are relative to performance of the policy that best mimics the oracle on the training data. In difficult decision-making problems, however, it can be hard to find a good policy that has a low training error, since the oracle's policy may resides in a space that is not imitable in the learner's policy space. For instance, the task loss function can be highly non-convex in the learner's parameter space and very different from the surrogate loss.

When the optimal action is hard to achieve, we propose to coach the learner with easy-to-learn actions and let it gradually approach the oracle (Section 3). A coach trains the learner iteratively in a fashion similar to DAgger. At each iteration it demonstrates actions that the learner's current policy prefers *and* have a small task loss. The coach becomes harsher by showing more oracle actions as the learner makes progress. Intuitively, this allows the learner to move towards a better action without much effort. Thus our algorithm achieves the best action gradually instead of aiming at an impractical goal from the beginning. We analyze our algorithm by a reduction to online learning and show that our approach achieves a lower regret bound than DAgger that uses the oracle action (Section 3.1). Our method is also related to direct loss minimization [4] for structured prediction and methods of selecting oracle translations in machine translation [5, 6] (Section 5).

Our approach is motivated by a formulation of budgeted learning as a sequential decision-making problem [7, 8] (Section 4). In this setting, features are acquired at a cost, such as computation time and experiment expense. In dynamic feature selection, we would like to sequentially select a subset of features for *each instance* at *test time* according to a user-specified accuracy-cost trade-off. Experimental results show that coaching has a more stable training curve and achieves lower task loss than state-of-the-art imitation learning algorithms.

Our major contribution is a meta-algorithm for hard imitation learning tasks where the available policy space is not adequate for imitating the oracle. Our main theoretical result is Theorem 4 which states that coaching as a smooth transition from the learner to the oracle have a lower regret bound than only using the oracle.

## 2   Background

In a sequential decision-making problem, we have a set of *states* $S$, a set of *actions* $A$ and a policy space $\Pi$. An agent follows a *policy* $\pi\colon S \rightarrow A$ that determines which action to take in a given state. After taking action $a$ in state $s$, the environment responds by some immediate loss $L(s, a)$. We assume $L(s, a)$ is bounded in $[0, 1]$. The agent is then taken to the next state $s'$ according to the transition probability $P(s'|s, a)$. We denote $d_\pi^t$ the state distribution at time $t$ after executing $\pi$ from time 1 to $t-1$, and $d_\pi$ the average state distribution of states over $T$ steps. Then the $T$-step expected loss of $\pi$ is $J(\pi) = \sum_{t=1}^{T} \mathbb{E}_{s \sim d_\pi^t}[L(s, \pi(s)) = T\mathbb{E}_{s \sim d_\pi}[L(s, \pi(s))]$. A *trajectory* is a complete sequence of $\langle s, a, L(s, a) \rangle$ tuples from the starting state to a goal state. Our goal is to learn a policy $\pi \in \Pi$ that minimizes the *task loss* $J(\pi)$. We assume that $\Pi$ is a closed, bounded and non-empty convex set in Euclidean space; a policy $\pi$ can be parameterized by a vector $\boldsymbol{w} \in \mathbb{R}^d$.

In imitation learning, we define an oracle that executes policy $\pi^*$ and demonstrates actions $a_s^* = \arg\min_{a \in A} L(s, a)$ in state $s$. The learner only attempts to imitate the oracle's behavior without any notion of the task loss function. Thus minimizing the task loss is reduced to minimizing a *surrogate loss* with respect to the oracle's policy.

### 2.1   Imitation by Classification

A typical approach to imitation learning is to use the oracle's trajectories as supervised data and learn a policy (multiclass classifier) that predicts the oracle action under distribution of states induced by running the oracle's policy. At each step $t$, we collect a training example $(s_t, \pi^*(s_t))$, where $\pi^*(s_t)$ is the oracle's action (class label) in state $s_t$. Let $\ell(s, \pi, \pi^*(s))$ denote the surrogate loss of executing $\pi$ in state $s$ with respect to $\pi^*(s)$. This can be any convex loss function used for training the classifier, for example, hinge loss in SVM. Using any standard supervised learning algorithm, we can learn a policy

$$\hat{\pi} = \arg\min_{\pi \in \Pi} \mathbb{E}_{s \sim d_{\pi^*}}[\ell(s, \pi, \pi^*(s))]. \tag{1}$$

We then bound $J(\hat{\pi})$ based on how well the learner imitates the oracle. Assuming $\ell(s, \pi, \pi^*(s))$ is an upper bound on the 0-1 loss and $L(s, a)$ is bounded in [0,1], Ross and Bagnell [9] have shown that:

**Theorem 1.** *Let $\mathbb{E}_{s \sim d_{\pi^*}}[\ell(s, \hat{\pi}, \pi^*(s))] = \epsilon$, then $J(\hat{\pi}) \leq J(\pi^*) + T^2\epsilon$.*

One drawback of the supervised approach is that it ignores the fact that the state distribution is different for the oracle and the learner. When the learner cannot mimic the oracle perfectly (i.e. classification error occurs), the wrong action will change the following state distribution. Thus the learned policy is not able to handle situations where the learner follows a wrong path that is never chosen by the oracle, hence the quadratically increasing loss. In fact in the worst case, performance can approach random guessing, even for arbitrarily small $\epsilon$ [10].

Ross *et al.* [3] generalized Theorem 1 to any policy that has $\epsilon$ surrogate loss under its *own* state distribution, i.e. $\mathbb{E}_{s \sim d_\pi}[\ell(s, \pi, \pi^*(s))] = \epsilon$. Let $Q_t^{\pi'}(s, \pi)$ denote the $t$-step loss of executing $\pi$ in the initial state and then running $\pi'$. We have the following:

**Theorem 2.** *If $Q_{T-t+1}^{\pi^*}(s, \pi) - Q_{T-t+1}^{\pi^*}(s, \pi^*) \leq u$ for all action $a$, $t \in \{1, 2, \ldots, T\}$, then $J(\pi) \leq J(\pi^*) + uT\epsilon$.*

It basically says that when $\pi$ chooses a different action from $\pi^*$ at time step $t$, if the cumulative cost due to this error is bounded by $u$, then the relative task loss is $O(uT)$.

## 2.2 Dataset Aggregation

The above problem of insufficient exploration can be alleviated by iteratively learning a policy trained under states visited by both the oracle and the learner. For example, during training one can use a "mixture oracle" that at times takes an action given by the previous learned policy [11]. Alternatively, at each iteration one can learn a policy from trajectories generated by all previous policies [3].

In its simplest form, the Dataset Aggregation (DAgger) algorithm [3] works as follows. Let $s_\pi$ denote a state visited by executing $\pi$. In the first iteration, we collect a training set $\mathcal{D}_1 = \{(s_{\pi^*}, \pi^*(s_{\pi^*}))\}$ from the oracle ($\pi_1 = \pi^*$) and learn a policy $\pi_2$. This is the same as the supervised approach to imitation. In iteration $i$, we collect trajectories by executing the previous policy $\pi_i$ and form the training set $\mathcal{D}_i$ by labeling $s_{\pi_i}$ with the oracle action $\pi^*(s_{\pi_i})$; $\pi_{i+1}$ is then learned on $\mathcal{D}_1 \bigcup \ldots \mathcal{D}_i$. Intuitively, this enables the learner to make up for past failures to mimic the oracle. Thus we can obtain a policy that performs well under its own induced state distribution.

## 2.3 Reduction to Online Learning

Let $\ell_i(\pi) = \mathbb{E}_{s \sim d_{\pi_i}}[\ell(s, \pi, \pi^*(s))]$ denote the expected surrogate loss of executing $\pi$ in states distributed according to $d_{\pi_i}$. In an online learning setting, in iteration $i$ an algorithm executes policy $\pi_i$ and observes loss $\ell_i(\pi_i)$. It then provides a different policy $\pi_{i+1}$ in the next iteration and observes $\ell_{i+1}(\pi_{i+1})$. A no-regret algorithm guarantees that in $N$ iterations

$$\frac{1}{N} \sum_{i=1}^{N} \ell_i(\pi_i) - \min_{\pi \in \Pi} \frac{1}{N} \sum_{i=1}^{N} \ell_i(\pi) \leq \gamma_N \tag{2}$$

and $\lim_{N \to \infty} \gamma_N = 0$.

Assuming a strongly convex loss function, *Follow-The-Leader* is a simple no-regret algorithm. In each iteration it picks the policy that works best so far: $\pi_{i+1} = \underset{\pi \in \Pi}{\arg \min} \sum_{j=1}^{i} \ell_j(\pi)$. Similarly, in DAgger at iteration $i$ we choose the policy that has the minimum surrogate loss on all previous data. Thus it can be interpreted as *Follow-The-Leader* where trajectories collected in each iteration are treated as one online-learning example.

Assume that $\ell(s, \pi, \pi^*(s))$ is a strongly convex loss in $\pi$ upper bounding the 0-1 loss. We denote the sequence of learned policies $\pi_1, \pi_2, \ldots, \pi_N$ by $\pi_{1:N}$. Let $\epsilon_N = \min_{\pi \in \Pi} \frac{1}{N} \sum_{i=1}^{N} \mathbb{E}_{s \sim d_{\pi_i}}[\ell(s, \pi, \pi^*(s))]$ be the minimum loss we can achieve in the policy space $\Pi$. In the infinite sample per iteration case, following proofs in [3] we have:

**Theorem 3.** *For DAgger, if $N$ is $O(uT \log T)$ and $Q_{T-t+1}^{\pi^*}(s, \pi) - Q_{T-t+1}^{\pi^*}(s, \pi^*) \leq u$, there exists a policy $\pi \in \pi_{1:N}$ s.t. $J(\pi) \leq J(\pi^*) + uT\epsilon_N + O(1)$.*

This theorem holds for any no-regret online learning algorithm and can be generalized to the finite sample case as well.

# 3 Imitation by Coaching

An oracle can be hard to imitate in two ways. First, the learning policy space is far from the space that the oracle policy lies in, meaning that the learner only has limited learning ability. Second, the environment information known by the oracle cannot be sufficiently inferred from the state, meaning that the learner does not have access to good learning resources. In the online learning setting, a too-good oracle may result in adversarially varying loss functions over iterations from the learner's perspective. This may cause violent changes during policy updating. These difficulties result in a substantial gap between the oracle's performance and the best performance achievable in the policy space $\Pi$ (i.e. a large $\epsilon_N$ in Theorem 3).

**Algorithm 1** DAgger by Coaching

    Initialize $\mathcal{D} \leftarrow \emptyset$
    Initialize $\pi_1 \leftarrow \pi^*$
    **for** $i = 1$ **to** $N$ **do**
        Sample $T$-step trajectories using $\pi_i$
        Collect coaching dataset $\mathcal{D}_i = \left\{ (s_{\pi_i}, \arg\max_{a \in A} \lambda_i \cdot \text{score}_{\pi_i}(s_{\pi_i}, a) - L(s_{\pi_i}, a)) \right\}$
        Aggregate datasets $\mathcal{D} \leftarrow \mathcal{D} \bigcup \mathcal{D}_i$
        Train policy $\pi_{i+1}$ on $\mathcal{D}$
    **end for**
    **Return** best $\pi_i$ evaluated on validation set

To address this problem, we define a coach in place of the oracle. To better instruct the learner, a coach should demonstrate actions that are not much worse than the oracle action but are easier to achieve within the learner's ability. The lower an action's task loss is, the closer it is to the oracle action. The higher an action is ranked by the learner's current policy, the more it is preferred by the learner, thus easier to learn. Therefore, similar to [6], we define a *hope action* that combines the task loss and the score of the learner's current policy. Let $\text{score}_{\pi_i}(s, a)$ be a measure of how likely $\pi_i$ chooses action $a$ in state $s$. We define $\tilde{\pi}_i$ by

$$\tilde{\pi}_i(s) = \arg\max_{a \in A} \lambda_i \cdot \text{score}_{\pi_i}(s, a) - L(s, a) \tag{3}$$

where $\lambda_i$ is a nonnegative parameter specifying how close the coach is to the oracle. In the first iteration, we set $\lambda_1 = 0$ as the learner has not learned any model yet. Algorithm 1 shows the training process. Our intuition is that when the learner has difficulty performing the optimal action, the coach should lower the goal properly and let the learner gradually achieving the original goal in a more stable way.

## 3.1 Theoretical Analysis

Let $\tilde{\ell}_i(\pi) = \mathbb{E}_{s \sim d_{\pi_i}}[\ell(s, \pi, \tilde{\pi}_i(s))]$ denote the expected surrogate loss with respect to $\tilde{\pi}_i$. We denote $\tilde{\epsilon}_N = \frac{1}{N} \min_{\pi \in \Pi} \sum_{i=1}^{N} \tilde{\ell}_i(\pi)$ the minimum loss of the best policy in hindsight with respect to hope actions. The main result of this paper is the following theorem:

**Theorem 4.** *For DAgger with coaching, if $N$ is $O(uT \log T)$ and $Q^{\pi^*}_{T-t+1}(s, \pi) - Q^{\pi^*}_{T-t+1}(s, \pi^*) \leq u$, there exists a policy $\pi \in \pi_{1:N}$ s.t. $J(\pi) \leq J(\pi^*) + uT\tilde{\epsilon}_N + O(1)$.*

It is important to note that both the DAgger theorem and the coaching theorem provide a relative guarantee. They depend on whether we can find a policy that has small training error in each *Follow-The-Leader* step. However, in practice, for hard learning tasks DAgger may fail to find such a good policy. Through coaching, we can always adjust $\lambda$ to create a more learnable oracle policy space, thus get a relatively good policy that has small training error, at the price of running a few more iterations.

To prove this theorem, we first derive a regret bound for coaching, and then follows the proofs of DAgger.

We consider a policy $\pi$ parameterized by a vector $\boldsymbol{w} \in \mathbb{R}^d$. Let $\phi \colon S \times A \to \mathbb{R}^d$ be a *feature map* describing the state. The predicted action is

$$\hat{a}_{\pi,s} = \arg\max_{a \in A} \boldsymbol{w}^T \phi(s, a) \tag{4}$$

and the hope action is

$$\tilde{a}_{\pi,s} = \arg\max_{a \in A} \lambda \cdot \boldsymbol{w}^T \phi(s, a) - L(s, a). \tag{5}$$

We assume that the loss function $\ell \colon \mathbb{R}^d \to \mathbb{R}$ is a convex upper bound of the 0-1 loss. Further, it can be written as $\ell(s, \pi, \pi^*(s)) = f(\boldsymbol{w}^T \phi(s, \pi(s)), \pi^*(s))$ for a function $f \colon \mathbb{R} \to \mathbb{R}$ and a feature vector $\|\phi(s, a)\|_2 \leq R$. We assume that $f$ is twice differentiable and convex in $\boldsymbol{w}^T \phi(s, \pi(s))$, which is common for most loss functions used by supervised classification methods.

It has been shown that given a strongly convex loss function $\ell$, *Follow-The-Leader* has $O(\log N)$ regret [12, 13]. More specifically, given the above assumptions we have:

**Theorem 5.** *Let* $D = \max_{\boldsymbol{w}_1, \boldsymbol{w}_2 \in \mathbb{R}^d} \|\boldsymbol{w}_1 - \boldsymbol{w}_2\|_2$ *be the diameter of the convex set* $\mathbb{R}^d$. *For some* $b, m > 0$, *assume that for all* $\boldsymbol{w} \in \mathbb{R}^d$, *we have* $|f'(\boldsymbol{w}^T \phi(s, a))| \leq b$ *and* $|f''(\boldsymbol{w}^T \phi(s, a))| \geq m$. *Then* Follow-The-Leader *on functions* $\ell$ *have the following regret:*

$$
\begin{aligned}
\sum_{i=1}^{N} \ell_i(\pi_i) - \min_{\pi \in \Pi} \sum_{i=1}^{N} \ell_i(\pi) &\leq \sum_{i=1}^{N} \ell_i(\pi_i) - \sum_{i=1}^{N} \ell_i(\pi_{i+1}) \\
&\leq \frac{2nb^2}{m} \left[ \log \left( \frac{DRmN}{b} \right) + 1 \right]
\end{aligned}
$$

To analyze the regret using surrogate loss with respect to hope actions, we use the following lemma:

**Lemma 1.** $\sum_{i=1}^{N} \ell_i(\pi_i) - \min_{\pi \in \Pi} \sum_{i=1}^{N} \tilde{\ell}_i(\pi) \leq \sum_{i=1}^{N} \ell_i(\pi_i) - \sum_{i=1}^{N} \tilde{\ell}_i(\pi_{i+1})$.

*Proof.* We prove inductively that $\sum_{i=1}^{N} \tilde{\ell}_i(\pi_{i+1}) \leq \min_{\pi \in \Pi} \sum_{i=1}^{N} \tilde{\ell}_i(\pi)$.

When $N = 1$, by *Follow-The-Leader* we have $\pi_2 = \arg \min_{\pi \in \Pi} \tilde{\ell}_1(\pi)$, thus $\tilde{\ell}_1(\pi_2) = \min_{\pi \in \Pi} \tilde{\ell}_1(\pi)$.

Assume correctness for $N - 1$, then

$$
\begin{aligned}
\sum_{i=1}^{N} \tilde{\ell}_i(\pi_{i+1}) &\leq \min_{\pi \in \Pi} \sum_{i=1}^{N-1} \tilde{\ell}_i(\pi) + \tilde{\ell}_N(\pi_{N+1}) \quad \text{(inductive assumption)} \\
&\leq \sum_{i=1}^{N-1} \tilde{\ell}_i(\pi_{N+1}) + \tilde{\ell}_N(\pi_{N+1}) = \min_{\pi \in \Pi} \sum_{i=1}^{N} \tilde{\ell}_i(\pi)
\end{aligned}
$$

The last equality is due to the fact that $\pi_{N+1} = \arg \min_{\pi \in \Pi} \sum_{i=1}^{N} \tilde{\ell}_i(\pi)$. $\qquad \square$

To see how learning from $\tilde{\pi}_i$ allows us to approaching $\pi^*$, we derive the regret bound of $\sum_{i=1}^{N} \ell_i(\pi_i) - \min_{\pi \in \Pi} \sum_{i=1}^{N} \tilde{\ell}_i(\pi)$.

**Theorem 6.** *Assume that* $\boldsymbol{w}_i$ *is upper bounded by* $C$, *i.e. for all* $i$ $\|\boldsymbol{w}_i\|_2 \leq C$, $\|\phi(s, a)\|_2 \leq R$ *and* $|L(s, a) - L(s, a')| \geq \epsilon$ *for some action* $a, a' \in A$. *Assume* $\lambda_i$ *is non-increasing and define* $n_\lambda$ *as the largest* $n < N$ *such that* $\lambda_{n_\lambda} \geq \frac{\epsilon}{2RC}$. *Let* $\ell_{\max}$ *be an upper bound on the loss, i.e. for all* $i$, $\ell_i(s, \pi_i, \pi^*(s)) \leq \ell_{\max}$. *We have*

$$
\sum_{i=1}^{N} \ell_i(\pi_i) - \min_{\pi \in \Pi} \sum_{i=1}^{N} \tilde{\ell}_i(\pi) \leq 2\ell_{\max} n_\lambda + \frac{2nb^2}{m} \left[ \log \left( \frac{DRmN}{b} \right) + 1 \right]
$$

*Proof.* Given Lemma 1, we only need to bound the RHS, which can be written as

$$
\left( \sum_{i=1}^{N} \ell_i(\pi_i) - \tilde{\ell}_i(\pi_i) \right) + \left( \sum_{i=1}^{N} \tilde{\ell}_i(\pi_i) - \tilde{\ell}_i(\pi_{i+1}) \right). \tag{6}
$$

To bound the first term, we consider a binary action space $A = \{1, -1\}$ for clarity. The proof can be extended to the general case in a straightforward manner.

Note that in states where $a_s^* = \tilde{a}_{\pi,s}$, $\ell(s, \pi, \pi^*(s)) = \ell(s, \pi, \tilde{\pi}(s))$. Thus we only need to consider situations where $a_s^* \neq \tilde{a}_{\pi,s}$:

$$
\begin{aligned}
&\ell_i(\pi_i) - \tilde{\ell}_i(\pi_i) \\
= \ & \mathbb{E}_{s \sim d_{\pi_i}} \left[ (\ell_i(s, \pi_i, -1) - \ell_i(s, \pi_i, 1)) \mathbf{1}_{\{s: \tilde{a}_{\pi_i, s} = 1, a_s^* = -1\}} \right] \\
&+ \mathbb{E}_{s \sim d_{\pi_i}} \left[ (\ell_i(s, \pi_i, 1) - \ell_i(s, \pi_i, -1)) \mathbf{1}_{\{s: \tilde{a}_{\pi_i, s} = -1, a_s^* = 1\}} \right]
\end{aligned}
$$

In the binary case, we define $\Delta L(s) = L(s, 1) - L(s, -1)$ and $\Delta\phi(s) = \phi(s, 1) - \phi(s, -1)$.

**Case 1** $\quad \tilde{a}_{\pi_i,s} = 1$ and $a_s^* = -1$.

$\tilde{a}_{\pi_i,s} = 1$ implies $\lambda_i \boldsymbol{w}_i^T \Delta\phi(s) \geq \Delta L(s)$ and $a_s^* = -1$ implies $\Delta L(s) > 0$. Together we have $\Delta L(s) \in (0, \lambda_i \boldsymbol{w}_i^T \Delta\phi(s)]$. From this we know that $\boldsymbol{w}_i^T \Delta\phi(s) \geq 0$ since $\lambda_i > 0$, which implies $\hat{a}_{\pi_i} = 1$. Therefore we have

$$
\begin{aligned}
& p(a_s^* = -1, \tilde{a}_{\pi_i,s} = 1, \hat{a}_{\pi_i,s} = 1) \\
=\ & p(\tilde{a}_{\pi_i,s} = 1 | a_s^* = -1, \hat{a}_{\pi_i,s} = 1) p(\hat{a}\pi_i, s = 1) p(a_s^* = -1) \\
=\ & p\left(\lambda_i \geq \frac{\Delta L(s)}{\boldsymbol{w}_i^T \Delta\phi(s)}\right) \cdot p(\boldsymbol{w}_i^T \Delta\phi(s) \geq 0) \cdot p(\Delta L(s) > 0) \\
\leq\ & p\left(\lambda_i \geq \frac{\epsilon}{2RC}\right) \cdot 1 \cdot 1 \ =\ p\left(\lambda_i \geq \frac{\epsilon}{2RC}\right)
\end{aligned}
$$

Let $n_\lambda$ be the largest $n < N$ such that $\lambda_i \geq \dfrac{\epsilon}{2RC}$, we have

$$
\sum_{i=1}^{N} \mathbb{E}_{s\sim d_{\pi_i}} \left[ (\ell_i(s, \pi_i, -1) - \ell_i(s, \pi_i, 1)) \, \mathbf{1}_{\{s:\ \tilde{a}_{\pi_i,s}=1, a_s^*=-1\}} \right] \leq \ell_{\max} n_\lambda
$$

For example, let $\lambda_i$ decrease exponentially, e.g., $\lambda_i = \lambda_0 e^{-i}$. If $\lambda_0 < \dfrac{\epsilon e^N}{2RC}$, Then $n_\lambda = \lceil \log \dfrac{2\lambda_0 RC}{\epsilon} \rceil$.

**Case 2** $\quad \tilde{a}_{\pi_i,s} = -1$ and $a_s^* = 1$. This is symmetrical to Case 1. Similar arguments yield the same bound.

In the online learning setting, imitating the coach is to observe the loss $\tilde{\ell}_i(\pi_i)$ and learn a policy $\pi_{i+1} = \arg\min_{\pi\in\Pi} \sum_{j=1}^{i} \tilde{\ell}_j(\pi)$ at iteration $i$. This is indeed equivalent to *Follow-The-Leader* except that we replaced the loss function. Thus Theorem 5 gives the bound of the second term.

Equation 6 is then bounded by $2\ell_{\max} n_\lambda + \dfrac{2nb^2}{m} \left[ \log\left(\dfrac{DRmN}{b}\right) + 1 \right]$. $\qquad \square$

Now we can prove Theorem 4. Consider the best policy in $\pi_{1:N}$, we have

$$
\begin{aligned}
\min_{\pi\in\pi_{1:N}} \mathbb{E}_{s\sim d_\pi}[\ell(s, \pi, \pi^*(s))] \ &\leq\ \frac{1}{N} \sum_{i=1}^{N} \mathbb{E}_{s\sim d_{\pi_i}}[\ell(s, \pi_i, \pi^*(s))] \\
&\leq\ \tilde{\epsilon}_N + \frac{2\ell_{\max} n_\lambda}{N} + \frac{2nb^2}{mN} \left[ \log\left(\frac{DRmN}{b}\right) + 1 \right]
\end{aligned}
$$

When $N$ is $\Omega(T \log T)$, the regret is $O(1/T)$. Applying Theorem 2 completes the proof.

## 4 Experiments

We apply imitation learning to a novel dynamic feature selection problem. We consider the setting where a pretrained model (*data classifier*) on a complete feature set is given and each feature has a known cost. At *test time*, we would like to dynamically select a subset of features for *each instance* and be able to explicitly specify the accuracy-cost trade-off. This can be naturally framed as a sequential decision-making problem. The state includes all features selected so far. The action space includes a set of non-selected features and the stop action. At each time step, the policy decides whether to stop acquiring features and make a prediction; if not, which feature(s) to purchase next. Achieving an accuracy-cost trade-off corresponds to finding the optimal policy minimizing a loss function. We define the loss function as a combination of accuracy and cost:

$$
L(s, a) = \alpha \cdot \text{cost}(s) - \text{margin}(a) \tag{7}
$$

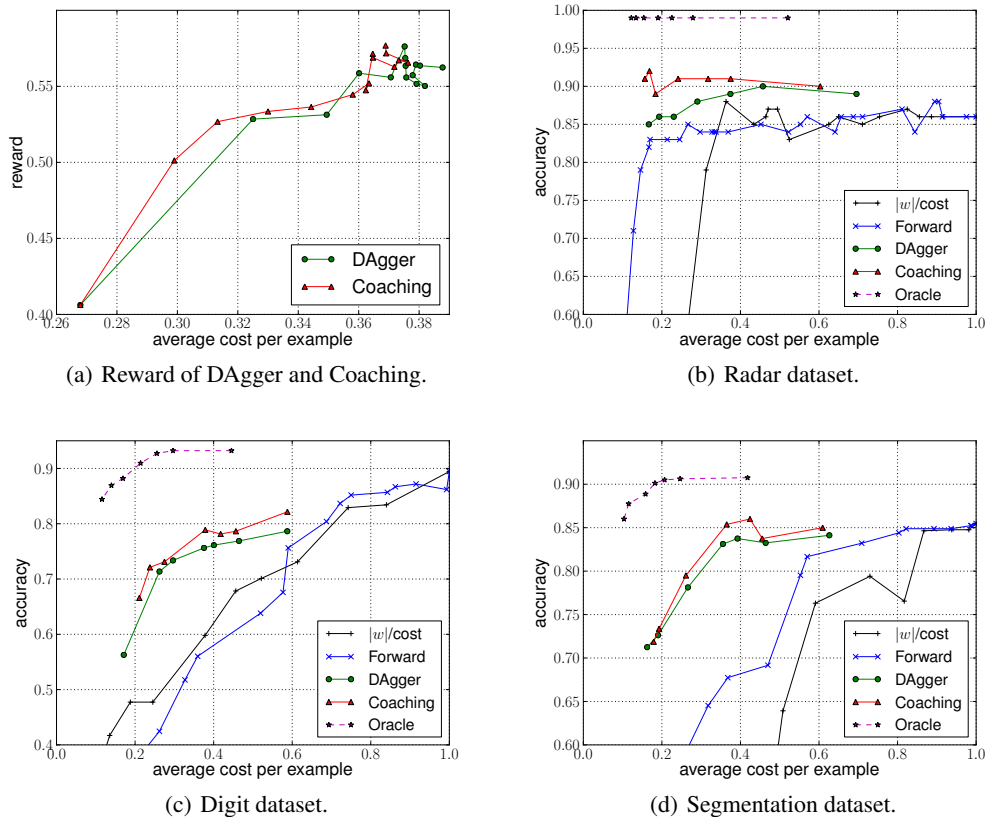

(a) Reward of DAgger and Coaching.

(b) Radar dataset.

(c) Digit dataset.

(d) Segmentation dataset.

Figure 1: 1(a) shows reward versus cost of DAgger and Coaching over 15 iterations on the digit dataset with $\alpha = 0.5$. 1(b) to 1(d) show accuracy versus cost on the three datasets. For DAgger and Coaching, we show results when $\alpha = 0, 0.1, 0.25, 0.5, 1.0, 1.5, 2$.

where margin$(a)$ denote the margin of classifying the instance after action $a$; cost$(s)$ denote the user-defined cost of all selected features in the current state $s$; and $\alpha$ is a user-specified trade-off parameter. Since we consider feature selection for each single instance here, the average margin reflects accuracy on the whole datasets.

## 4.1 Dynamic Feature Selection by Imitation Learning

Ideally, an oracle should lead to a subset of features having the maximum reward. However, we have too large a state space to exhaustedly search for the optimal subset of features. In addition, the oracle action may not be unique since the optimal subset of features do not have to be selected in a fixed order. We address this problem by using a forward-selection oracle. Given a state $s$, the oracle iterates through the action space and calculates each action's loss; it then chooses the action that leads to the minimum immediate loss in the current state. We define $\phi(s_t, a)$ as a concatenation of the current feature vector and a meta-feature vector that provides information about previous classification results and cost.

In most cases, our oracle can achieve high accuracy with rather small cost. Considering a linear classifier, as the oracle already knows the correct class label of an instance, it can simply choose, for example, a positive feature that has a positive weight to correctly classify a positive instance. In addition, at the start state even when $\phi(s_0, a)$ are almost the same for all instances, the oracle may tend to choose features that favor the instance's class. This makes the oracle's behavior very hard to imitate. In the next section we show that in this case coaching achieves better results than using an oracle.

### 4.2 Experimental Results

We perform experiments on three UCI datasets (radar signal, digit recognition, image segmentation). Random costs are assigned to features. We first compare the learning curve of DAgger and Coaching over 15 iterations on the digit dataset with $\alpha = 0.5$ in Figure 1(a). We can see that DAgger makes a big improvement in the second iteration, while Coaching takes smaller steps but achieves higher reward gradually. In addition, the reward of Coaching changes smoothly and grows stably, which means coaching avoids drastic change of the policy.

To test the effect of dynamic selection, we compare our results with DAgger and two static feature selection baselines that sequentially add features according to a ranked list. The first baseline (denoted by Forward) ranks features according to the standard forward feature selection algorithm without any notion of the cost. The second baseline (denoted by $|w|$/cost) uses a cost-sensitive ranking scheme based on $|w|$/cost, the weight of a feature divided by its cost. Therefore, features having high scores are expected to be cost-efficient. We give the results in Figure 1(b) to 1(d). To get results of our dynamic feature selection algorithm at different costs, we set $\alpha$ in the loss function to be 0.0, 0.1, 0.25, 0.5, 1.0, 1.5, 2.0 and use the best policy evaluated on the development set for each $\alpha$. For coaching, we set $\lambda_2 = 1$ and decrease it by $e^{-1}$ in each iteration. First, we can see that dynamically selecting features for each instance significantly improves the accuracy at a small cost. Sometimes, it even achieves higher accuracy than using all features. Second, we notice that there is a substantial gap between the learned policy's performance and the oracle's, however, in almost all settings Coaching achieves higher reward, i.e. higher accuracy at a lower cost as shown in the figures. Through coaching, we can reduce the gap by taking small steps towards the oracle. However, the learned policy is still much worse compared to the oracle's policy. This is because coaching is still inherently limited by the insufficient policy space, which can be fixed by using expensive kernels and nonlinear policies.

## 5 Related Work

The idea of using hope action is similar to what Chiang *et al.* [6] and Liang *et al.* [5] have used for selecting oracle translations in machine translation. They maximized a linear combination of the BLEU score (similar to negative task loss in our case) and the model score to find good translations that are easier to train against. More recently, McAllester *et al.* [4] defined the direct label that combines model score and task loss from a different view: they showed that using a perceptron-like training methods and update towards the direct label is equivalent to perform gradient descent on the task loss.

Coaching is also similar to proximal methods in online learning [14, 15]. They avoid large changes during updating and achieve the original goal gradually. In proximal regularization, we want the updated parameter vector to stay close to the previous one. Do *et al.* [14] showed that solving the original learning problem through a sequence of modified optimization tasks whose objectives have greater curvature can achieve a lower regret bound.

## 6 Conclusion and Future Work

In this paper, we consider the situation in imitation learning where an oracle's performance is far from what is achievable in the learning space. We propose a coaching algorithm that lets the learner target at easier goals first and gradually approaches the oracle. We show that coaching has a lower regret bound both theoretically and empirically. In the future, we are interested in formally defining the hardness of a problem so that we know exactly in which cases coaching is more suitable than DAgger. Another direction is to develop a similar coaching process in online convex optimization by optimizing a sequence of approximating functions. We are also interested in applying coaching to more complex structured prediction problems in natural language processing and computer vision.

## References

[1] P. Abbeel and A. Y. Ng. Apprenticeship learning via inverse reinforcement learning. In *ICML*, 2004.

[2] M. Veloso B. D. Argall, S. Chernova and B. Browning. A survey of robot learning from demonstration. 2009.

[3] Stéphane. Ross, Geoffrey J. Gordon, and J. Andrew. Bagnell. A reduction of imitation learning and structured prediction to no-regret online learning. In *AISTATS*, 2011.

[4] D. McAllester, T. Hazan, and J. Keshet. Direct loss minimization for structured prediction. In *NIPS*, 2010.

[5] D. Klein P. Liang, A. Bouchard-Ct and B. Taskar. An end-to-end discriminative approach to machine translation. In *ACL*, 2006.

[6] D. Chiang, Y. Marton, and P. Resnik. Online large-margin training of syntactic and structural translation features. In *EMNLP*, 2008.

[7] R. Busa-Fekete D. Benbouzid and B. Kégl. Fast classification using space decision dags. In *ICML*, 2012.

[8] P. Preux G. Dulac-Arnold, L. Denoyer and P. Gallinari. Datum-wise classification: a sequential approach to sparsity. In *ECML*, 2011.

[9] Stéphane Ross and J. Andrew Bagnell. Efficient reductions for imitation learning. In *AISTATS*, 2010.

[10] Kääriäinen. Lower bounds for reductions. In *Atomic Learning Workshop*, 2006.

[11] Hal Daumé III, John Langford, and Daniel Marcu. Search-based structured prediction. *Machine Learning Journal (MLJ)*, 2009.

[12] Elad Hazan, Adam Kalai, Satyen Kale, and Amit Agarwal. Logarithmic regret algorithms for online convex optimization. In *COLT*, pages 499–513, 2006.

[13] Sham M. Kakade and Shai Shalev-shwartz. Mind the duality gap: Logarithmic regret algorithms for online optimization. In *NIPS*, 2008.

[14] Q. Le C. B. Do and C.S. Foo. Proximal regularization for online and batch learning. In *ICML*, 2009.

[15] H Brendan Mcmahan. Follow-the-regularized-leader and mirror descent : Equivalence theorems and l1 regularization. *JMLR*, 15:525–533, 2011.

